# Learning Kernels with Radiuses of Minimum Enclosing Balls

**Kun Gai**  **Guangyun Chen**  **Changshui Zhang**

State Key Laboratory on Intelligent Technology and Systems
Tsinghua National Laboratory for Information Science and Technology (TNList)
Department of Automation, Tsinghua University, Beijing 100084, China
`{gaik02, cgy08}@mails.thu.edu.cn, zcs@mail.thu.edu.cn`

## Abstract

In this paper, we point out that there exist scaling and initialization problems in most existing multiple kernel learning (MKL) approaches, which employ the large margin principle to jointly learn both a kernel and an SVM classifier. The reason is that the margin itself can not well describe how good a kernel is due to the negligence of the scaling. We use the ratio between the margin and the radius of the minimum enclosing ball to measure the goodness of a kernel, and present a new minimization formulation for kernel learning. This formulation is invariant to scalings of learned kernels, and when learning linear combination of basis kernels it is also invariant to scalings of basis kernels and to the types (e.g., $L_1$ or $L_2$) of norm constraints on combination coefficients. We establish the differentiability of our formulation, and propose a gradient projection algorithm for kernel learning. Experiments show that our method significantly outperforms both SVM with the uniform combination of basis kernels and other state-of-art MKL approaches.

## 1 Introduction

In the past years, kernel methods, like support vector machines (SVM), have achieved great success in many learning problems, such as classification and regression. For such tasks, the performance strongly depends on the choice of the kernels used. A good kernel function, which implicitly characterizes a suitable transformation of input data, can greatly benefit the accuracy of the predictor. However, when there are many available kernels, it is difficult for the user to pick out a suitable one.

Kernel learning has been developed to jointly learn both a kernel function and an SVM classifier. Chapelle et al. [1] present several principles to tune parameters in kernel functions. In particular, when the learned kernel is restricted to be a linear combination of multiple basis kernels, the problem of learning the combination coefficients as well as an SVM classifier is usually called multiple kernel learning (MKL). Lanckriet et al. [2] formulate the MKL problem as a quadratically constrained quadratic programming problem, which implicitly uses an $L_1$ norm constraint to promote sparse combinations. To enhance the computational efficiency, different approaches for solving this MKL problem have been proposed using SMO-like strategies [3], semi-infinite linear program [4], gradient-based methods [5], and second-order optimization [6]. Some other subsequent work explores more generality of multiple kernel learning by promoting non-sparse [7, 8] or group-sparse [9] combinations of basis kernels, or using other forms of learned kernels, e.g., a combination of an exponential number of kernels [10] or nonlinear combinations [11, 12, 13].

Most existing MKL approaches employ the objective function used in SVM. With an acceptable empirical loss, they aim to find the kernel which leads to the largest margin of the SVM classifier. However, despite the substantial progress in both the algorithmic design and the theoretical understanding for the MKL problem, none of the approaches seems to reliably outperform baseline

methods, like SVM with the uniform combination of basis kernels [13]. As will be shown in this paper, the large margin principle used in these methods causes the scaling problem and the initialization problem, which can strongly affect final solutions of learned kernels as well as performances. It implicates that the large margin preference can not reliably result in a good kernel, and thus the margin itself is not a suitable measure of the goodness of a kernel.

Motivated by the generalization bounds for SVM and kernel learning, we use the ratio between the margin of the SVM classifier and the radius of the minimum enclosing ball (MEB) of data in the feature space endowed with the learned kernel as a measure of the goodness of the kernel, and propose a new kernel learning formulation. Our formulation differs from the radius-based principle by Chapelle et al. [1]. Their principle is sensitive to kernel scalings when a nonzero empirical loss is allowed, also causing the same problems as the margin-based formulations. We prove that our formulation is invariant to scalings of learned kernels, and also invariant to initial scalings of basis kernels and to the types (e.g., $L_1$ or $L_2$) of norm constraints on kernel parameters for the MKL problem. Therefore our formulation completely addresses the scaling and initialization problems. Experiments show that our approach gives significant performance improvements both over SVM with the uniform combination of basis kernels and over other state-of-art kernel learning methods.

Our proposed kernel learning problem can be reformulated to a tri-level optimization problem. We establish the differentiability of a general family of multilevel optimization problems. This enables us to generally tackle the radius of the minimal enclosing ball, or other complicated optimal value functions, in the kernel learning framework by simple gradient-based methods. We hope that our results will also benefit other learning problems.

The paper is structured as follows. Section 2 shows problems in previous MKL formulations. In Section 3 we present a new kernel learning formulation and give discussions. Then, we study the differentiability of multilevel optimization problems and give an efficient algorithm in Section 4 and Section 5, respectively. Experiments are shown in Section 6. Finally, we close with a conclusion.

## 2 Measuring how good a kernel is

Let $\mathcal{D} = \{(x_1, y_1), ..., (x_n, y_n)\}$ denote a training set of $n$ pairs of input points $x_i \in \mathcal{X}$ and target labels $y_i \in \{\pm 1\}$. Suppose we have a kernel family $\mathcal{K} = \{k : \mathcal{X} \times \mathcal{X} \to \mathbb{R}\}$, in which any kernel function $k$ implicitly defines a transformation $\phi(\cdot; k)$ from the input space $\mathcal{X}$ to a feature space by $k(x_c, x_d) = \langle \phi(x_c; k), \phi(x_d; k) \rangle$. Let a classifier be linear in the feature space endowed with $k$, as

$$f(x; w, b, k) = \langle \phi(x; k), w \rangle + b, \tag{1}$$

the sign of which is used to classify data. The task of kernel learning (for binary classification) is to learn both a kernel function $k \in \mathcal{K}$ and a classifier $w$ and $b$.

To make the problem trackable, the learned kernel is usually restricted to a parametric form $k^{(\theta)}(\cdot, \cdot)$, where $\theta = [\theta_i]_i$ is the kernel parameter. Then the problem of learning a kernel transfers to the problem of learning a kernel parameter $\theta$. The most common used kernel form is a linear combination of multiple basis kernels, as

$$k^{(\theta)}(\cdot, \cdot) = \sum_{j=1}^{m} \theta_j k_j(\cdot, \cdot), \quad \theta_j \geq 0. \tag{2}$$

### 2.1 Problems in multiple kernel learning

Most existing MKL approaches, e.g., [2, 4, 5], employ the equivalent objective function as in SVM:

$$\min_{k,w,b,\xi_i} \quad \frac{1}{2}\|w\|^2 + C\sum_i \xi_i, \quad \text{s.t.} \quad y_i f(x_i; w, b, k) + \xi_i \geq 1, \ \xi_i \geq 0, \tag{3}$$

where $\xi_i$ is the hinge loss. This problem can be reformulated to

$$\min_k : \quad \tilde{G}(k), \tag{4}$$

where $\quad \tilde{G}(k) = \min_{w,b,\xi_i} \frac{1}{2}\|w\|^2 + C\sum_i \xi_i, \quad \text{s.t.} \quad y_i f(x_i; w, b, k) + \xi_i \geq 1, \ \xi_i \geq 0. \tag{5}$

For any kernel $k$, the optimal classifier $w$ and $b$ is actually the SVM classifier with the kernel $k$. Let $\gamma$ denote the margin of the SVM classifier in the feature space endowed with $k$. We have $\gamma^{-2} = \|w\|^2$. Thus the term $\|w\|^2$ makes formulation (3) prefer the kernel that results in an SVM classifier with a larger margin (as well as an acceptable empirical loss). Here, a natural question is that for different kernels whether the margins of SVM classifiers can well measure the goodness of the kernels.

To answer this question, we consider what happens when a kernel $k$ is enlarged by a scalar $a$: $k^{new} = ak$, where $a > 1$. The corresponding transformations satisfy $\phi(\cdot; k^{new}) = \sqrt{a}\phi(\cdot; k)$. For $k$, let $\{w^*, b^*\}$ denote the optimal solution of (5). For $k^{new}$, we set $w_2 = w_1^*/\sqrt{a}$ and $b_2 = b_1^*$, then we have $\|w_2\|^2 = \|w_1^*\|^2/a$, and $f(x; w_2, b_2, k^{new})$ and $f(x; w_1^*, b_1^*, k)$ are the same classifier, resulting in the same $\xi_i$. Then we obtain: $\tilde{G}(ak) = \tilde{G}(k^{new}) \leq \frac{1}{2}\|w_2\|^2 + C\sum_i \xi_i < \frac{1}{2}\|w_1^*\|^2 + C\sum_i \xi_i = \tilde{G}(k)$, which means the enlarged kernel gives a larger margin and a smaller objective value. As a consequence, on one hand, the large margin preference guides the scaling of the learned kernel to be as large as possible. On the other hand, any kernel, even the one resulting in a bad performance, can give an arbitrarily large margin by enlarging its scaling. This problem is called *the scaling problem*. It shows that the margin is not a suitable measure of the goodness of a kernel.

In the linear combination case, the scaling problem causes that the kernel parameter $\theta$ does not converge in the optimization. A remedy is to use a norm constraint on $\theta$. However, it has been shown in recent literature [7, 9] that different types of norm constraints fit different data sets. So users face the difficulty of choosing a suitable norm constraint. Even after a norm constraint is selected, the scaling problem also causes another problem about the initialization. Consider an $L_1$ norm constraint and a learned kernel which is a combination of two basis kernels, as

$$k^{(\theta)}(\cdot, \cdot) = \theta_1 k_1(\cdot, \cdot) + \theta_2 k_2(\cdot, \cdot), \quad \theta_1, \theta_2 \geq 0, \quad \theta_1 + \theta_2 = 1. \tag{6}$$

To leave the empirical loss out of consideration, assume: (a) both $k_1$ and $k_2$ can lead to zero empirical loss, (b) $k_1$ results in a larger margin than $k_2$. For simplicity, we further restrict $\theta_1$ and $\theta_2$ to be equal to 0 or 1, to enable kernel selection. The MKL formulation (3), of course, will choose $k_1$ from $\{k_1, k_2\}$ due to the large margin preference. Then we set $k_1^{\text{new}}(\cdot, \cdot) = ak_1(\cdot, \cdot)$, where $a$ is a small scalar to make that $k_1^{\text{new}}$ has a smaller margin than $k_2$. After $k_1^{\text{new}}$ substitutes for $k_1$, the MKL formulation (3) will select $k_2$ from $\{k_1^{\text{new}}, k_2\}$. The example shows that the final solution can be greatly affected by the initial scalings of basis kernels, although a norm constraint is used. This problem is called *the initialization problem*. When the MKL framework is extended from the linear combination cases to the nonlinear cases, the scaling problem becomes more serious, as even a finite scaling of the learned kernel may not be generally guaranteed by a simple norm constraint on kernel parameters for some kernel forms. These problems implicate that the margin itself is not enough to measure the goodness of kernels.

## 2.2 Measuring the goodness of kernels with the radiuses of MEB

Now we need to find a more reasonable way to measure the goodness of kernels. Below we introduce the generalization error bounds for SVM and kernel learning, which inspire us to consider the minimum enclosing ball to learn a kernel. For SVM with a fixed kernel, it is well known that the estimation error, which denotes the gap between the expected error and the empirical error, is bounded by $\sqrt{\mathcal{O}(R^2\gamma^{-2})/n}$, where $R$ is the radius of the minimum enclosing ball (MEB) of data in the feature space endowed with the kernel used. For SVM with a kernel learned from a kernel family $\mathcal{K}$, if we restrict that the radius of the minimum enclosing ball in the feature space endowed with the learned kernel to be no larger than $R$, then the theoretical results of Srebro and Ben-David [14] say: for any fixed margin $\gamma > 0$ and any fixed radius $R > 0$, with probability at least $1 - \delta$ over a training set of size $n$, the estimation error is no larger than $\sqrt{\frac{8}{n}(2 + d_\phi \log \frac{128en^3R^2}{\gamma^2 d_\phi} + 256\frac{R^2}{\gamma^2}\log \frac{en\gamma}{8R}\log \frac{128nR^2}{\gamma^2} - \log\delta)}$. Scalar $d_\phi$ denotes the pseudodimension [14] of the kernel family $\mathcal{K}$. For example, $d_\phi$ of linear combination kernels is no larger than the number of basis kernels, and $d_\phi$ of the Gaussian kernels with a form of $k^{(\theta)}(x^a, x^b) = e^{-\theta\|x^a - x^b\|^2}$ is no larger than 1 (See [14] for more details). The above results clearly state that the generalization error bounds for SVM with both fixed kernels and learned kernels depend on the ratio between the margin $\gamma$ and the radius $R$ of the minimum enclosing ball of data. Although some new results of the generalization bounds for kernel learning, like [15], give different types of dependencies on $d_\phi$, they also rely on the margin-and-radius ratio.

In SVM with a fixed kernel, the radius $R$ is a constant and we can safely minimize $\|w\|^2$ (as well as the empirical loss). However, in kernel learning, the radius $R$ changes drastically from one kernel to another (An example is given in the supplemental materials: when we uniformly combine $p$ basis kernels by $k_{\text{unif}} = \sum_{j=1}^p \frac{1}{p}k_j$, the squared radius becomes only $\frac{1}{p}$ of the squared radius of each basis kernel.). Thus we should also take the radius into account. As a result, we use the ratio between the margin $\gamma$ and the radius $R$ to measure how good a kernel is for kernel learning.

Given any kernel $k$, the radius of the minimum enclosing ball, denoted by $R(k)$, can be obtained by:

$$R^2(k) = \min_{y,c} \ y, \quad \text{s.t.} \quad y \geq \|\phi(x_i; k) - c\|^2. \tag{7}$$

This problem is a convex minimization problem, being equivalent to its dual problem, as

$$R^2(k) = \ \max_{\beta_i} \ \sum_i \beta_i k(x_i, x_i) - \sum_{i,j} \beta_i k(x_i, x_j)\beta_j, \quad \text{s.t.} \ \sum_i \beta_i = 1, \ \beta_i \geq 0, \tag{8}$$

which shows a property of $R^2(k)$: for any kernel $k$ and any scalar $a > 0$, we have $R^2(ak) = aR^2(k)$.

## 3 Learning kernels with the radiuses

Considering the ratio between the margin and the radius of MEB, we propose a new formulation, as

$$\min_{k,w,b,\xi_i} \ \tfrac{1}{2}R^2(k)\|w\|^2 + C\sum_i \xi_i, \quad \text{s.t.} \quad y_i(\langle\phi(x_i; k), w\rangle + b) + \xi_i \geq 1, \ \xi_i \geq 0, \tag{9}$$

where $R^2(k)\|w\|^2$ is a radius-based regularizer that prefers a large ratio between the margin and the radius, and $\sum_i \xi_i$ is the hinge loss which is an upper bound of empirical misclassified error. This optimization problem is called radius based kernel learning problem, referred to as *RKL*.

Chapelle et al. [1] also utilize the radius of MEB to tune kernel parameters for hard margin SVM. Our formulation (9) is equivalent to theirs if $\xi_i$ is restricted to be zero. To give a soft margin version, they modify the kernel matrix $K(\theta) = K(\theta) + \frac{1}{C}I$, resulting in a formulation equivalent to:

$$\min_{\theta,w,b,\xi_i} \ \tfrac{1}{2}R^2(k^{(\theta)})\|w\|^2 + CR^2(k^{(\theta)})\sum_i \xi_i^2, \ \text{s.t.} \ y_i(\langle\phi(x_i; k^{(\theta)}), w\rangle + b) + \xi_i \geq 1, \ \xi_i \geq 0. \tag{10}$$

The function $R^2(k^{(\theta)})$ in the second term, which may become small, makes that minimizing the objective function can not reliably give a small empirical loss, even when $C$ is large. Besides, when we reduce the scaling of a kernel by multiplying it with a small scalar $a$ and substitute $\tilde{w} = w/\sqrt{a}$ for $w$ to keep the same $\xi_i$, the objective function always decreases (due to the decrease of $R^2$ in the empirical loss term), still leading to scaling problems. Do et al. [16] recently propose to learn a linear kernel combination, as defined in (2), through

$$\min_{\theta,w_j,b,\xi_i} \ \tfrac{1}{2}\sum_j \frac{\|w_j\|^2}{\theta_j} + \frac{C}{\sum_j \theta_j R^2(k_j)}\sum_i \xi_i^2, \ \text{s.t.} \ y_i(\sum_j \langle w_j, \phi(x_i; k_j)\rangle + b) + \xi_i \geq 1, \ \xi_i \geq 0. \tag{11}$$

Their objective function also can be always decreased by multiplying $\theta$ with a large scalar. Thus their method does not address the scaling problem, also resulting in the initialization problem. If we initially adjust the scalings of basis kernels to make each $R(k_j)$ be equal to each other, then their formulation is equivalent to the margin-based formulation (3). Different from the above formulations, our formulation (9) is invariant to scalings of kernels.

### 3.1 Invariance to scalings of kernels

Now we discuss the properties of formulation (9). The RKL problem can be reformulated to

$$\min_k \ G(k), \tag{12}$$

where $G(k) = \min_{w,b,\xi_i} \ \tfrac{1}{2}R^2(k)\|w\|^2 + C\sum_i \xi_i, \ \text{s.t.} \ y_i(\langle\phi(x_i; k), w\rangle + b) + \xi_i \geq 1, \ \xi_i \geq 0. \tag{13}$

Functional $G(k)$ defines a measure of the goodness of kernel functions, which consider a trade-off between the margin-and-radius ratio and the empirical loss. This functional is invariant to the scaling of $k$, as stated by the following proposition.

**Proposition 1.** *For any kernel $k$ and any scalar $a > 0$, equation $G(ak) = G(k)$ holds.*

*Proof.* For the scaled kernel $ak$, equation $R^2(ak) = aR^2(k)$ holds. Thereby, we get

$$G(ak) = \min_{w,b,\xi_i} \ \tfrac{a}{2}R^2(k)\|w\|^2 + C\sum_i \xi_i, \ \text{s.t.} \ y_i(\langle\sqrt{a}\phi(x_i; k), w\rangle + b) + \xi_i \geq 1, \ \xi_i \geq 0. \tag{14}$$

Let $\frac{\tilde{w}}{\sqrt{a}} = w$ replace $w$ in (14), and then (14) becomes equivalent to (13). Thus $G(ak) = G(k)$. $\qquad\square$

For a parametric kernel form $k^{(\theta)}$, the RKL problem transfers to minimizing a function $g(\theta) \doteq G(k^{(\theta)})$. Here we temporarily focus on the linear combination case defined by (2), and use $g_{\text{linear}}(\theta)$ to denote $g(\theta)$ in such case. Due to the scaling invariance, for any $\theta$ and any $a > 0$, we have $g_{\text{linear}}(a\theta) = g_{\text{linear}}(\theta)$. It makes the problem of minimizing $g_{\text{linear}}(\theta)$ be invariant to the types of norm constraints on $\theta$, as stated in the following.

**Proposition 2.** *Given any norm definition $\mathcal{N}(\cdot)$ and any set $\mathcal{S} \subseteq \mathbb{R}$, suppose there exists $c > 0$ that satisfies $c \in \mathcal{S}$. Let (a) denote the problem of minimizing $g_{\mathrm{linear}}(\theta)$ s.t. $\theta_i \geq 0$, and (b) denote the problem of minimizing $g_{\mathrm{linear}}(\theta)$ s.t. $\theta_i \geq 0$ and $\mathcal{N}(\theta) \in \mathcal{S}$. Then we have: (1) For any local (global) optimal solution of (a), denoted by $\theta^a$, $\frac{c}{\mathcal{N}(\theta^a)}\theta^a$ is also the local (global) optimal solution of (b). (2) For any local (global) optimal solution of (b), denoted by $\theta^b$, $\theta^b$ is also the local (global) optimal solution of (a).*

*Proof.* The complete proof is given in the the supplemental materials. Here we only prove the equivalence of global optimal solutions of (a) and (b). On one hand, if $\theta^a$ is the global optimal solution of (a), then for any $\theta$ that satisfies $\theta_i \geq 0$ and $\mathcal{N}(\theta) \in \mathcal{S}$, we have $g_{\mathrm{linear}}(\frac{c}{\mathcal{N}(\theta)}\theta^a) = g_{\mathrm{linear}}(\theta^a) \leq g(\theta)$. Due to $\mathcal{N}(\frac{c}{\mathcal{N}(\theta^a)}\theta^a) = c \in \mathcal{S}$, $\frac{c}{\mathcal{N}(\theta)}\theta^a$ also satisfies the constraint of (b), and thus $\frac{c}{\mathcal{N}(\theta)}\theta^a$ is the global optimal solution of (b). On the other hand, for any $\theta$ ($\theta_i \geq 0$), $g_{\mathrm{linear}}(\frac{c}{\mathcal{N}(\theta)}\theta) = g_{\mathrm{linear}}(\theta)$ due to the scaling invariance. If $\theta^b$ is the global optimal solution of (b), then for any $\theta$ ($\theta_i \geq 0$), as $\frac{c}{\mathcal{N}(\theta)}\theta$ satisfies the constraint of (b), we have $g_{\mathrm{linear}}(\theta^b) \leq g_{\mathrm{linear}}(\frac{c}{\mathcal{N}(\theta)}\theta)$, giving $g_{\mathrm{linear}}(\theta^b) \leq g_{\mathrm{linear}}(\theta)$. Thus $\theta^b$ is the global optimal solution of (a). $\qquad\square$

As the problems of minimizing $g_{\mathrm{linear}}(\theta)$ under different types of norm constraints on $\theta$ are all equivalent to the same problem without any norm constraint, they are equivalent to each other. Based on the above proposition, we can also get the another conclusion: in the linear combination case the minimization problem (12) is also invariant to the initial scalings of basis kernels (see below).

**Proposition 3.** *Let $k_j$ denote basis kernels, and $a_j > 0$ be initial scaling coefficients of basis kernels. Give a norm constraint $\mathcal{N}(\theta) \in \mathcal{S}$, which is by the same definition as in Proposition 2. Let (a) denote the problem of minimizing $G(\sum_j \theta_j k_j)$ w.r.t. $\theta$ s.t. $\theta_i \geq 0$ and $\mathcal{N}(\theta) \in \mathcal{S}$, and (b) denote the problem with different initial scalings: minimizing $G(\sum_j \theta_j a_j k_j)$ w.r.t. $\theta$ s.t. $\theta_i \geq 0$ and $\mathcal{N}(\theta) \in \mathcal{S}$. Then: (1) Problem (a) and problem (b) have the same local and global optimums. (2) For any local (global) optimal solution of (b), denoted by $\theta^b$, $[\frac{ca_j\theta_j^b}{\mathcal{N}([a_t\theta_t^b]_t)}]_j$ is also the local (global) optimal solution of (a).*

*Proof.* By proposition 2, problems (b) is equivalent to the one without any norm constraint: minimizing $G(\sum_j \theta_j a_j k_j)$ w.r.t. $\theta$ s.t. $\theta_i \geq 0$, which is denoted by problem (c). Let $\tilde{\theta}_j = a_j \theta_j$, and then problem (c) is equivalent to the problem of minimizing $G(\sum_j \tilde{\theta}_j k_j)$ w.r.t. $\tilde{\theta}$ s.t. $\tilde{\theta}_i \geq 0$, which is denoted by problem (d) (local and global optimal solutions of problems (c) and (d) have one-to-one correspondences due to the simple transform $\tilde{\theta}_j = a_j \theta_j$). Again, by Proposition 2, problem (d) is equivalent to the one with $\mathcal{N}(\theta) \in \mathcal{S}$, which is indeed problem (a). So we have conclusion (1). By proper transformations of optimal solutions of these equivalent problems, we get conclusion (2). $\quad\square$

Note that in Proposition 3, optimal solutions of problems (a) and (b), which are with different initial scalings of basis kernels, actually result in the same kernel combinations up to the scalings.

As shown in the above three propositions, our proposed formulation not only completely addresses scaling and initialization problems, but also is not sensitive to the types of norm constraints used.

### 3.2 Reformulation to a tri-level optimization problem

The remaining task is to optimize the RKL problem (12). Given a parametric kernel form $k^{(\theta)}$, for any parameter $\theta$, to obtain the value of the objective function $g(\theta) = G(k^{(\theta)})$ in (12), we need to solve the SVM-like problem in (13), which is a convex minimization problem and can be solved by its dual problem. Indeed, the whole RKL problem is transformed to a tri-level optimization problem:

$$\min_\theta g(\theta), \tag{15}$$

where $g(\theta) = \left\{ \max_{\alpha_i} \sum_i \alpha_i - \frac{1}{2r^2(\theta)} \sum_{i,j} \alpha_i \alpha_j y_i y_j K_{i,j}(\theta), \text{ s.t. } \sum_i \alpha_i y_i = 0,\ 0 \leq \alpha_i \leq C \right\},$ (16)

where $r^2(\theta) = \left\{ \max_{\beta_i} \sum_i \beta_i K_{i,j}(\theta) - \sum_{i,j} \beta_i K_{i,j}(\theta)\beta_j, \text{ s.t. } \sum_i \beta_i = 1,\ \beta_i \geq 0 \right\}.$ (17)

Notation $K(\theta)$ denotes the kernel matrix $[k^{(\theta)}(x_i, x_j)]_{i,j}$. The above formulations show that given any $\theta$ the calculation of a value of $g(\theta)$ requires solving a bi-level optimization problem. First, solve the MEB dual problem (17), and obtain the optimal value $r^2(\theta)$ and the optimal solution, denoted by

$\beta_i^*$. Then, take $r^2(\theta)$ into the objective function of the SVM dual problem (16), solve it, and obtain the value of $g(\theta)$, as well as the optimal solution of (16), denoted by $\alpha_i^*$. Unlike in other kernel learning approaches, here the optimization of the SVM dual problem relies on another optimal value function $r^2(\theta)$, making the RKL problem more challenging.

If $g(\theta)$, which is the objective function in the top-level optimization, is differentiable and we can get its derivatives, then we can use a variety of gradient-based methods to solve the RKL problem. So in next section, we study the differentiability of a general family of multilevel optimization problems.

## 4 Differentiability of the multilevel optimization problem

The Danskin's theorem [17] states the differentiability of the optimal value of a single-level optimization problem, and has been applied in many MKL algorithms, e.g., [5, 12]. Unfortunately, it is not directly applicable to the optimal value of a multilevel optimization problem. Below we generalize the Danskin's theorem and give new results about the multilevel optimization problem.

Let $Y$ be a metric space, and $X$, $U$ and $Z$ be normed spaces. Suppose: (1) The function $g_1(x, u, z)$, is continuous on $X \times U \times Z$. (2) For all $x \in X$ the function $g_1(x, \cdot, \cdot)$ is continuously differentiable. (3) The function $g_2(y, x, u)$ $(g_2 : Y \times X \times U \to Z)$ is continuous on $Y \times X \times U$. (4) For all $y \in Y$ the function $g_2(y, \cdot, \cdot)$ is continuously differentiable. (5) Sets $\Phi_X \subseteq X$ and $\Phi_Y \subseteq Y$ are compact. By these notations, we propose the following theorem about bi-level optimal value functions.

**Theorem 1.** *Let us define a bi-level optimal value function as*

$$v_1(u) = \inf_{x \in \Phi_X} g_1(x, u, v_2(x, u)), \tag{18}$$

*where $v_2(x, u)$ is another optimal value function as*

$$v_2(x, u) = \inf_{y \in \Phi_Y} g_2(y, x, u). \tag{19}$$

*If for any $x$ and $u$, $g_2(\cdot, x, u)$ has a unique minimizer $y^*(x, u)$ over $\Phi_Y$, then $y^*(x, u)$ are continuous on $X \times U$, and $v_1(u)$ is directionally differentiable. Furthermore, if for any $u$, the $g_1(\cdot, u, v_2(\cdot, u))$ has also a unique minimizer $x^*(u)$ over $\Phi_X$, then*

1. *the minimizer $x^*(u)$ are continuous on $U$,*

2. *$v_1(u)$ is continuously differentiable, and its derivative is equal to*

$$\frac{dv_1(u)}{du} = \left( \frac{\partial g_1(x^*, u, v_2)}{\partial u} + \frac{\partial v_2(x^*, u)}{\partial u} \frac{\partial g_1(x^*, u, v_2)}{\partial v_2} \right)\Big|_{v_2 = v_2(x^*, u)}, \quad \text{where } \frac{\partial v_2(x^*, u)}{\partial u} = \frac{\partial g_2(y^*, x^*, u)}{\partial u}. \tag{20}$$

The proof is given in supplemental materials. To apply Theorem 1 to the objective function $g(\theta)$ in the RKL problem (15), we shall make sure the following two conditions are satisfied. First, both the MEB dual problem (17) and the SVM dual problem (16) must have unique optimal solutions. This can be guaranteed by that the kernel matrix $K(\theta)$ is strictly positive definite. Second, the kernel matrix $K(\theta)$ shall be continuously differentiable to $\theta$. Both conditions can be met in the linear combination case when each basis kernel matrix is strictly positive definite, and can also be easily satisfied in nonlinear cases, like in [11, 12]. If these two conditions are met, then $g(\theta)$ is continuously differentiable and

$$\frac{dg(\theta)}{d\theta} = -\frac{1}{2r^2(\theta)} \sum_{i,j} \alpha_i^* \alpha_j^* y_i y_j \frac{dK_{i,j}(\theta)}{d\theta} + \frac{1}{2r^4(\theta)} \sum_{i,j} \alpha_i^* \alpha_j^* y_i y_j K_{i,j}(\theta) \frac{dr^2(\theta)}{d\theta}, \tag{21}$$

where $\alpha_i^*$ is the optimal solution of the SVM dual problem (16), and

$$\frac{dr^2(\theta)}{d\theta} = \sum_i \beta_i^* \frac{dK_{i,i}(\theta)}{d\theta} - \sum_{i,j} \beta_i^* \frac{dK_{i,j}(\theta)}{d\theta} \beta_j^*, \tag{22}$$

where $\beta_i^*$ is the optimal solution of the MEB dual problem (17). In above equations, the value of $\frac{dK_{i,j}(\theta)}{d\theta}$ is needed. It depends on the specific form of the parametric kernels, and the deriving of it is easy. For example, for the linear combination kernel $K_{i,j}(\theta) = \sum_m \theta_m K_{i,j}^m$, we have $\frac{\partial K_{i,j}(\theta)}{\partial \theta_m} = K_{i,j}^m$. For the Gaussian kernel $K_{i,j}(\theta) = e^{-\theta \|x_i - x_j\|^2}$, we have $\frac{dK_{i,j}(\theta)}{d\theta} = -K_{i,j}(\theta)\|x_i - x_j\|^2$.

## 5 Algorithm

With the derivative of $g(\theta)$, we use the standard gradient projection approach with the Armijo rule [18] for selecting step sizes to address the RKL problem. To compare with the most popular kernel learning algorithm, simpleMKL [5], in experiments we employ the linear combination

kernel form with nonnegative combination coefficients, as defined in (2). In addition, we also consider three types of norm constraints on kernel parameters (combination coefficients): $L_1$, $L_2$ and no norm constraint. The $L_1$ and $L_2$ norm constraints are as $\sum_j \theta_j = 1$ and $\sum_j \theta_j^2 = 1$, respectively. The projection for the $L_1$ norm and nonnegative constraints can be efficiently done by the method of Duchi et al. [19]. The projection for only nonnegative constraints can be accomplished by setting negative elements to be zero. The projection for the $L_2$ norm and nonnegative constraints need another step after eliminating negative values: normalize $\theta$ by multiplying it with $\|\theta\|_2^{-1}$.

In our gradient projection algorithm, each calculation of the objective functions $g(\theta)$ needs solving an MEB problem (17) and an SVM problem (16), whereas the gradient calculation and projection steps have ignorable time complexity compared to MEB and SVM solvers. The MEB and SVM problems have similar forms of objective functions and constraints, and both of them can be efficiently solved by SMO algorithms. Moreover, previous solutions $\alpha_i^*$ and $\beta_i^*$ can be used as "hotstart" to accelerate the solvers. It is because optimal solutions of two problems are continuous to kernel parameter $\theta$ according to Theorem 1. Thus when $\theta$ moves a small step, the optimal solutions also will only change a little. In real experiments our approach usually achieves approximate convergence within one or two dozens of invocations of SVM and MEB solvers (For lack of space, examples of the convergence speed of our algorithm are shown in the supplemental materials).

In linear combination cases, the RKL problem, as the radius-based formulation by Chapelle et al. [1], is not convex. Gradient-based methods only guarantee local optimums. The following states the nontrivial quality of local optimal solutions and their connections to related convex problems.

**Proposition 4.** *In linear combination cases, for any local optimal solution of the RKL problem, denoted by $\theta^*$, there exist $C_1 > 0$ and $C_2 > 0$ that $\theta^*$ is the global optimal solution of the following convex problem:*

$$\min_{\theta, w_j, b, \xi_i} \frac{1}{2} \sum_j \|w_j\|^2 + C_1 r^2(\theta) + C_2 \sum_i \xi_i^2, \text{ s.t. } y_i (\sum_j \langle w_j, \phi(x_i; \theta_j k_j) \rangle + b) + \xi_i \geq 1, \ \xi_i \geq 0. \ (23)$$

The proof can be found in the supplemental materials. The proposition also gives another possible way to address the RKL problem: iteratively solve the convex problem (23) with a search for $C_1$ and $C_2$. However, it is difficult to find exact values of $C_1$ and $C_2$ by a grid search, and even a rough search will result in too high computational load. Besides, such method is also lack of extension ability to nonlinear parametric kernel forms. Then, in the experiments, we demonstrate that the gradient-based approach can give satisfactory performances, which are significantly better than ones of SVM with the uniform combination of basis kernels and of other kernel learning approaches.

## 6 Experiments

In this section, we illustrate the performances of our presented RKL approach, in comparison with SVM with the uniform combination of basis kernels (Unif), the margin-based MKL method using formulation (3) (MKL), and the kernel learning principle by Chapelle et al. [1] using formulation (10) (KL-C). The evaluation is made on eleven public available data sets from UCI repository [20] and LIBSVM Data [21] (see Table 1). All data sets have been normalized to be zero-means and unit-variances on every feature. The used basis kernels are the same as in SimpleMKL [5]: 10 Gaussian kernels with bandwiths $\gamma_G \in \{0.5, 1, 2, 5, 7, 10, 12, 15, 17, 20\}$ and 10 polynomial kernels of degree 1 to 10. All kernel matrices have been normalized to unit trace, as in [5, 7]. Note that although our RKL formulation is theoretically invariant to the initial scalings, the normalization is still applied in RKL to avoid numerical problems caused by large value kernel matrices in SVM and MEB solvers. To show impacts of different norm constraints, we use three types of them: $L_1$, $L_2$ and no norm constraint. With no norm constraint, only RKL can converge, and so only its results are reported. The SVM toolbox used is LIBSVM [21]. MKL with the $L_1$ norm constraint is solved by the code from SimpleMKL [5]. Other problems are solved by standard gradient-projection methods, where the calculation of gradients of the MKL formulation (3) and Chapelle's formulation (10) is the same as in [5] and [1], respectively. The initial $\theta$ is set to be $\frac{1}{20}e$, where $e$ is an all-ones vector.

The trade-off coefficients $C$ in SVM, MKL, KL-C and RKL are automatically determined by 3-fold cross-validations on training sets. In all methods, $C$ is selected from the set $S_{coef} \doteq \{0.01, 0.1, 1, 10, 100\}$. For each data set, we split it to five parts, and each time we use four parts as the training set and the remaining one as the test set. The average accuracies with standard deviations and average numbers of selected basis kernels are reported in Table 1.

Table 1: The testing accuracies (Acc.) with standard deviations (in parentheses), and the average numbers of selected basis kernels (Nk). We set the numbers of our method to be bold if our method outperforms both Unif and other two kernel learning approaches under the same norm constraint.

| Index | 1 | | 2 | | 3 | | 4 | | 5 | | 6 | | 7 | | 8 | |
|---|---|---|---|---|---|---|---|---|---|---|---|---|---|---|---|---|
| | Unif | | MKL | | KL-C | | Ours | | MKL | | KL-C | | Ours | | Ours | |
| Constraint | | | $L_1$ | | $L_1$ | | $L_1$ | | $L_2$ | | $L_2$ | | $L_2$ | | No | |
| Data set | Acc. | Nk | Acc. | Nk | Acc. | Nk | Acc. | Nk | Acc. | Nk | Acc. | Nk | Acc. | Nk | Acc. | Nk |
| Ionosphere | 94.0(1.4) | 20 | 92.9(1.6) | 3.8 | 86.0(1.9) | 4.0 | **95.7**(0.9) | 2.8 | 94.3(1.5) | 20 | 84.4(1.6) | 18 | **95.7**(0.9) | 3.0 | 95.7(0.9) | 3.0 |
| Splice | 51.7(0.1) | 20 | 79.5(1.9) | 1.0 | 80.5(1.9) | 2.8 | **86.5**(2.4) | 3.2 | 82.0(2.2) | 20 | 74.0(2.6) | 14 | **86.5**(2.4) | 2.2 | 86.3(2.5) | 3.2 |
| Liver | 58.0(0.0) | 20 | 59.1(1.4) | 4.2 | 62.9(3.5) | 4.0 | **64.1**(4.2) | 3.6 | 67.0(3.8) | 20 | 64.1(3.9) | 11 | 64.1(4.2) | 8.0 | 64.3(4.3) | 6.6 |
| Fourclass | 81.2(1.9) | 20 | 97.7(1.2) | 7.0 | 94.0(1.2) | 2.0 | **100** (0.0) | 1.0 | 97.3(1.6) | 20 | 94.0(1.3) | 17 | **100** (0.0) | 1.0 | 100 (0.0) | 1.6 |
| Heart | 83.7(6.1) | 20 | 84.1(5.7) | 7.4 | 83.3(5.9) | 1.8 | 84.1(5.7) | 5.2 | 83.7(5.8) | 20 | 83.3(5.1) | 19 | **84.4**(5.9) | 5.4 | 84.8(5.0) | 5.8 |
| Germannum | 70.0(0.0) | 20 | 70.0(0.0) | 7.2 | 71.9(1.8) | 9.8 | **73.7**(1.6) | 4.8 | 71.5(0.8) | 20 | 71.6(2.1) | 13 | **73.9**(1.2) | 6.0 | 73.9(1.8) | 5.8 |
| Musk1 | 61.4(2.9) | 20 | 85.5(2.9) | 1.6 | 73.9(2.9) | 2.0 | **93.3**(2.3) | 4.0 | 87.4(3.0) | 20 | 61.9(3.1) | 19 | **93.5**(2.2) | 3.8 | 93.3(2.3) | 3.8 |
| Wdbc | 94.4(1.8) | 20 | 97.0(1.8) | 1.2 | 97.4(2.3) | 4.6 | 97.4(1.6) | 6.2 | 96.8(1.6) | 20 | 97.4(2.0) | 11 | **97.6**(1.9) | 5.8 | 97.6(1.9) | 5.8 |
| Wpbc | 76.5(2.9) | 20 | 76.5(2.9) | 7.2 | 52.2(5.9) | 9.6 | 76.5(2.9) | 17 | 75.9(1.8) | 20 | 51.0(6.6) | 17 | **76.5**(2.9) | 15 | 76.5(2.9) | 15 |
| Sonar | 76.5(1.8) | 20 | 82.3(5.6) | 2.6 | 80.8(5.8) | 7.4 | **86.0**(2.6) | 2.6 | 85.2(2.9) | 20 | 80.2(5.9) | 11 | **86.0**(2.6) | 2.6 | 86.0(3.3) | 3.0 |
| Coloncancer | 67.2(11) | 20 | 82.6(8.5) | 13 | 74.5(4.4) | 11 | **84.2**(4.2) | 7.2 | 76.5(9.0) | 20 | 76.0(3.6) | 15 | **84.2**(4.2) | 5.6 | 84.2(4.2) | 7.6 |

The results in Table 1 can be summarized as follows. (a) RKL gives the best results on most sets. Under $L_1$ norm constraints, RKL (Index 4) outperforms all other methods (Index 1, 2, 3) on 8 out of 11 sets, and also gives results equal to the best ones of other methods on the remaining 3 sets. In particular, RKL gains 5 or more percents of accuracies on Splice, Liver and Musk1 over MKL, and gains more than 9 percents on four sets over KL-C. Under $L_2$ norm constraints, the results are similar: RKL (Index 7) outperforms other methods (Index 5, 6) on 10 out of 11 sets, with only 1 inverse result. (b) Both MKL and KL-C are sensitive to the types of norm constraints (Compare Index 2 and 5, as well as 3 and 6). As shown in recent literature [7, 9], for the MKL formulation, different types of norm constraints fit different data sets. However, RKL outperforms MKL (as well as KL-C) under both $L_1$ and $L_2$ norm constraints on most sets. (c) RKL is invariant to the types of norm constraints. See Index 4, 7 and 8. Most accuracy numbers of them are the same. Several exceptions with slight differences are possibly due to precisions of numerical computation. (d) For MKL, the $L_1$ norm constraint always results in sparse combinations, whereas the $L_2$ norm constraint always gives non-sparse results (see Index 2 and 5). (e) An interesting thing is that, our presented RKL gives sparse solutions on most sets, whatever types of norm constraints are used. As there usually exist redundancies in the basis kernels, the searching for good kernels and small empirical loss often directly leads to sparse solutions. We notice that KL-C under $L_2$ norm constraints also slightly promotes sparsity (Index 6). Compared to KL-C under $L_2$ norm constraints, RKL provides not only higher performances but also more sparsity, which benefits both interpretability and computational efficiency in prediction.

# 7 Conclusion

In this paper, we show that the margin term used in previous MKL formulations is not a suitable measure of the goodness of kernels, resulting in scaling and initialization problems. We propose a new formulation, called RKL, which uses the ratio between the margin and the radius of MEB to learn kernels. We prove that our formulation is invariant to kernel scalings, and also invariant to scalings of basis kernels and to the types of norm constraints for the MKL problem. Then, by establishing the differentiability of a general family of multilevel optimal value functions, we propose a gradient-based algorithm to address the RKL problem. We also provide the property of solutions of our algorithm. The experiments validate that our approach outperforms both SVM with the uniform combination of basis kernels and other state-of-art kernel learning methods.

**Acknowledgments**

The work is supported by the National Natural Science Foundation of China (NSFC) (Grant Nos. 60835002 and 61075004) and the National Basic Research Program (973 Program) (No. 2009CB320602).

# References

[1] O. Chapelle, V. Vapnik, O. Bousquet, and S. Mukherjee. Choosing multiple parameters for support vector machines. *Machine Learning*, 46(1):131–159, 2002.

[2] G.R.G. Lanckriet, N. Cristianini, P. Bartlett, L.E. Ghaoui, and M.I. Jordan. Learning the kernel matrix with semidefinite programming. *The Journal of Machine Learning Research*, 5:27–72, 2004.

[3] F.R. Bach, G.R.G. Lanckriet, and M.I. Jordan. Multiple kernel learning, conic duality, and the smo algorithm. In *Proceedings of the twenty-first international conference on Machine learning (ICML 2004)*, 2004.

[4] S. Sonnenburg, G. Rätsch, and C. Schäfer. A general and efficient multiple kernel learning algorithm. In *Adv. Neural. Inform. Process Syst. (NIPS 2005)*, 2006.

[5] A. Rakotomamonjy, F. Bach, S. Canu, and Y. Grandvalet. SimpleMKL. *Journal of Machine Learning Research*, 9:2491–2521, 2008.

[6] O. Chapelle and A. Rakotomamonjy. Second order optimization of kernel parameters. In *Proc. of the NIPS Workshop on Kernel Learning: Automatic Selection of Optimal Kernels*, 2008.

[7] M. Kloft, U. Brefeld, S. Sonnenburg, P. Laskov, K. Müller, and A. Zien. Efficient and Accurate lp-Norm Multiple Kernel Learning. In *Adv. Neural. Inform. Process Syst. (NIPS 2009)*, 2009.

[8] C. Cortes, M. Mohri, and A. Rostamizadeh. L2 regularization for learning kernels. In *Uncertainty in Artificial Intelligence*, 2009.

[9] J. Saketha Nath, G. Dinesh, S. Raman, Chiranjib Bhattacharyya, Aharon Ben-Tal, and K. R. Ramakrishnan. On the algorithmics and applications of a mixed-norm based kernel learning formulation. In *Adv. Neural. Inform. Process Syst. (NIPS 2009)*, 2009.

[10] F. Bach. Exploring large feature spaces with hierarchical multiple kernel learning. In *Adv. Neural. Inform. Process Syst. (NIPS 2008)*, 2008.

[11] M. Gönen and E. Alpaydin. Localized multiple kernel learning. In *Proceedings of the 25th international conference on Machine learning (ICML 2008)*, 2008.

[12] M. Varma and B.R. Babu. More generality in efficient multiple kernel learning. In *Proceedings of the 26th International Conference on Machine Learning (ICML 2009)*, 2009.

[13] C. Cortes, M. Mohri, and A. Rostamizadeh. Learning Non-Linear Combinations of Kernels. In *Adv. Neural. Inform. Process Syst. (NIPS 2009)*, 2009.

[14] N. Srebro and S. Ben-David. Learning bounds for support vector machines with learned kernels. In *Proceedings of the International Conference on Learning Theory (COLT 2006)*, pages 169–183. Springer, 2006.

[15] Yiming Ying and Colin Campbell. Generalization bounds for learning the kernel. In *Proceedings of the International Conference on Learning Theory (COLT 2009)*, 2009.

[16] H. Do, A. Kalousis, A. Woznica, and M. Hilario. Margin and Radius Based Multiple Kernel Learning. In *Proceedings of the European Conference on Machine Learning (ECML 2009)*, 2009.

[17] J.M. Danskin. The theory of max-min, with applications. *SIAM Journal on Applied Mathematics*, pages 641–664, 1966.

[18] Dimitri P. Bertsekas. *Nonlinear Programming*. Athena Scientific, Belmont, MA, September 1999.

[19] John Duchi, Shai Shalev-Shwartz, Yoram Singer, and Tushar Chandra. Efficient projections onto the l1-ball for learning in high dimensions. In *Proceedings of the 25th international conference on Machine learning (ICML 2008)*, 2008.

[20] A. Asuncion and D.J. Newman. UCI machine learning repository, 2007. Software available at http://www.ics.uci.edu/~mlearn/MLRepository.html.

[21] Chih-Chung Chang and Chih-Jen Lin. *LIBSVM: a library for support vector machines*, 2001. Software available at http://www.csie.ntu.edu.tw/~cjlin/libsvm.

